# Inference with Minimal Communication: a Decision-Theoretic Variational Approach

**O. Patrick Kreidl and Alan S. Willsky**
Department of Electrical Engineering and Computer Science
MIT Laboratory for Information and Decision Systems
Cambridge, MA 02139
{opk,willsky}@mit.edu

## Abstract

Given a directed graphical model with binary-valued hidden nodes and real-valued noisy observations, consider deciding upon the maximum a-posteriori (MAP) or the maximum posterior-marginal (MPM) assignment under the restriction that each node broadcasts only to its children exactly one single-bit message. We present a variational formulation, viewing the processing rules local to all nodes as degrees-of-freedom, that minimizes the loss in expected (MAP or MPM) performance subject to such online communication constraints. The approach leads to a novel message-passing algorithm to be executed *offline*, or before observations are realized, which mitigates the performance loss by iteratively coupling all rules in a manner implicitly driven by global statistics. We also provide (i) illustrative examples, (ii) assumptions that guarantee convergence and efficiency and (iii) connections to active research areas.

## 1   Introduction

Given a probabilistic model with discrete-valued hidden variables, Belief Propagation (BP) and related graph-based algorithms are commonly employed to solve for the Maximum A-Posteriori (MAP) assignment (i.e., the mode of the joint distribution of all hidden variables) and Maximum-Posterior-Marginal (MPM) assignment (i.e., the modes of the marginal distributions of every hidden variable) [1]. The established "message-passing" interpretation of BP extends naturally to a distributed network setting: associating to each node and edge in the graph a distinct processor and communication link, respectively, the algorithm is equivalent to a sequence of purely-local computations interleaved with only nearest-neighbor communications. Specifically, each computation event corresponds to a node evaluating its local *processing rule*, or a function by which all messages received in the preceding communication event map to messages sent in the next communication event.

Practically, the viability of BP appears to rest upon an implicit assumption that network communication resources are abundant. In a general network, because termination of the algorithm is in question, the required communication resources are a-priori unbounded. Even when termination can be guaranteed, transmission of exact messages presumes communication channels with infinite capacity (in bits per observation), or at least of sufficiently high bandwidth such that the resulting finite message precision is essentially error-free. In

some distributed settings (e.g., energy-limited wireless sensor networks), it may be prohibitively costly to justify such idealized online communications. While recent evidence suggests substantial but "small-enough" message errors will not alter the behavior of BP [2], [3], it also suggests BP may perform poorly when communication is very constrained.

Assuming communication constraints are severe, we examine the extent to which alternative processing rules can avoid a loss in (MAP or MPM) performance. Specifically, given a directed graphical model with binary-valued hidden variables and real-valued noisy observations, we assume each node may broadcast only to its children a single binary-valued message. We cast the problem within a variational formulation [4], seeking to minimize a decision-theoretic penalty function subject to such online communication constraints. The formulation turns out to be an extension of the optimization problem underlying the decentralized detection paradigm [5], [6], which advocates a team-theoretic [7] relaxation of the original problem to both justify a particular finite parameterization for all local processing rules and obtain an iterative algorithm to be executed *offline* (i.e., before observations are realized). To our knowledge, that this relaxation permits analytical progress given any directed acyclic network is new. Moreover, for MPM assignment in a tree-structured network, we discover an added convenience with respect to the envisioned distributed processor setting: the offline computation itself admits an efficient message-passing interpretation.

This paper is organized as follows. Section 2 details the decision-theoretic variational formulation for discrete-variable assignment. Section 3 summarizes the main results derived from its connection to decentralized detection, culminating in the offline message-passing algorithm and the assumptions that guarantee convergence and maximal efficiency. We omit the mathematical proofs [8] here, focusing instead on intuition and illustrative examples. Closing remarks and relations to other active research areas appear in Section 4.

## 2 Variational Formulation

In abstraction, the basic ingredients are (i) a joint distribution $p(x, y)$ for two length-$N$ random vectors $X$ and $Y$, taking hidden and observable values in the sets $\{0, 1\}^N$ and $\mathbb{R}^N$, respectively; (ii) a decision-theoretic penalty function $J : \Gamma \to \mathbb{R}$, where $\Gamma$ denotes the set of all candidate strategies $\gamma : \mathbb{R}^N \to \{0, 1\}^N$ for posterior assignment; and (iii) the set $\Gamma^{\mathcal{G}} \subset \Gamma$ of strategies that also respect stipulated communication constraints in a given $N$-node directed acyclic network $\mathcal{G}$. The ensuing optimization problem is expressed by

$$J(\gamma^*) \quad = \quad \min_{\gamma \in \Gamma} J(\gamma) \quad \text{subject to } \gamma \in \Gamma^{\mathcal{G}}, \tag{1}$$

where $\gamma^*$ then represents an *optimal network-constrained strategy* for discrete-variable assignment. The following subsections provide details unseen at this level of abstraction.

### 2.1 Decision-Theoretic Penalty Function

Let $U = \gamma(Y)$ denote the decision process induced from the observation process $Y$ by any candidate assignment strategy $\gamma \in \Gamma$. If we associate a numeric "cost" $c(u, x)$ to every possible joint realization of $(U, X)$, then the expected cost is a well-posed penalty function:

$$J(\gamma) = E\left[c\left(\gamma(Y), X\right)\right] = E\left[E\left[c(\gamma(Y), X) \mid Y\right]\right]. \tag{2}$$

Expanding the inner expectation and recognizing $p(x|y)$ to be proportional to $p(x)p(y|x)$ for every $y$ such that $p(y) > 0$, it follows that $\bar{\gamma}^*$ minimizes (2) over $\Gamma$ if and only if

$$\bar{\gamma}^*(Y) = \arg \min_{u \in \{0,1\}^N} \sum_{x \in \{0,1\}^N} p(x)c(u, x)p(Y|x) \quad \text{with probability one.} \tag{3}$$

Of note are (i) the likelihood function $p(Y|x)$ is a finite-dimensional sufficient statistic of $Y$, (ii) real-valued coefficients $\bar{b}(u, x)$ provide a finite parameterization of the function space $\Gamma$ and (iii) optimal coefficient values $\bar{b}^*(u, x) = p(x)c(u, x)$ are computable offline.

Before introducing communication constraints, we illustrate by examples how the decision-theoretic penalty function relates to familiar discrete-variable assignment problems.

*Example 1:* Let $c(u, x)$ indicate whether $u \neq x$. Then (2) and (3) specialize to, respectively, the *word error rate* (viewing each $x$ as an $N$-bit word) and the MAP strategy:

$$\bar{\gamma}^*(Y) = \arg \max_{x \in \{0,1\}^N} p(x|Y) \quad \text{with probability one.}$$

*Example 2:* Let $c(u, x) = \sum_{n=1}^N c_n(u_n, x_n)$, where each $c_n$ indicates whether $u_n \neq x_n$. Then (2) and (3) specialize to, respectively, the *bit error rate* and the MPM strategy:

$$\bar{\gamma}^*(Y) = \left( \arg \max_{x_1 \in \{0,1\}} p(x_1|Y), \ldots, \arg \max_{x_N \in \{0,1\}} p(x_N|Y) \right) \quad \text{with probability one.}$$

## 2.2 Network Communication Constraints

Let $\mathcal{G}(\mathcal{V}, \mathcal{E})$ be any directed acyclic graph with vertex set $\mathcal{V} = \{1, \ldots, N\}$ and edge set

$$\mathcal{E} = \{(i,j) \in \mathcal{V} \times \mathcal{V} \mid i \in \pi(j) \Leftrightarrow j \in \chi(i)\},$$

where index sets $\pi(n) \subset \mathcal{V}$ and $\chi(n) \subset \mathcal{V}$ indicate, respectively, the parents and children of each node $n \in \mathcal{V}$. Without loss-of-generality, we assume the node labels respect the natural partial-order implied by the graph $\mathcal{G}$; specifically, we assume every node $n$ has parent nodes $\pi(n) \subset \{1, \ldots, n-1\}$ and child nodes $\chi(n) \subset \{n+1, \ldots, N\}$. Local to each node $n \in \mathcal{V}$ are the respective components $X_n$ and $Y_n$ of the joint process $(X, Y)$. Under best-case assumptions on $p(x,y)$ and $\mathcal{G}$, Belief Propagation methods (e.g., max-product in Example 1, sum-product in Example 2) require at least $2|\mathcal{E}|$ real-valued messages per observation $Y = y$, one per direction along each edge in $\mathcal{G}$. In contrast, we insist upon a single forward-pass through $\mathcal{G}$ where each node $n$ broadcasts to its children (if any) a single binary-valued message. This yields communication overhead of only $|\mathcal{E}|$ bits per observation $Y = y$, but also renders the minimizing strategy of (3) infeasible.

Accepting that performance-communication tradeoffs are inherent to distributed algorithms, we proceed with the goal of minimizing the *loss* in performance relative to $J(\bar{\gamma}^*)$. Specifically, we now translate the stipulated restrictions on communication into explicit constraints on the function space $\Gamma$ over which to minimize (2). The simplest such translation assumes the binary-valued message produced by node $n$ also determines the respective component $u_n$ in decision vector $u = \gamma(y)$. Recognizing that every node $n$ receives the messages $u_{\pi(n)}$ from its parents (if any) as side information to $y_n$, any function of the form $\gamma_n : \mathbb{R} \times \{0,1\}^{|\pi(n)|} \to \{0,1\}$ is a feasible *processing rule*; we denote the set of all such rules by $\Gamma_n$. Then, every strategy in the set $\Gamma^{\mathcal{G}} = \Gamma_1 \times \cdots \times \Gamma_N$ respects the constraints.

# 3 Summary of Main Results

As stated in Section 1, the variational formulation presented in Section 2 can be viewed as an extension of the optimization problem underlying decentralized Bayesian detection [5], [6]. Even for specialized network structures (e.g., the $N$-node chain), it is known that exact solution to (1) is NP-hard, stemming from the absence of a guarantee that $\gamma^* \in \Gamma^{\mathcal{G}}$ possesses a finite parameterization. Also known is that analytical progress can be made for a relaxation of (1), which is based on the following intuition: if strategy $\gamma^* = (\gamma_1^*, \ldots, \gamma_N^*)$ is optimal over $\Gamma^{\mathcal{G}}$, then for each $n$ and assuming all components $i \in \mathcal{V} \backslash n$ are fixed at rules $\gamma_i^*$, the component rule $\gamma_n^*$ must be optimal over $\Gamma_n$. Decentralized detection has roots in team decision theory [7], a subset of game theory, in which the relaxation is named *person-by-person* (pbp) optimality. While global optimality always implies pbp-optimality, the converse is false—in general, there can be multiple pbp-optimal solutions with varying

penalty. Nonetheless, pbp-optimality (along with a specialized observation process) justifies a particular finite parameterization for the function space $\Gamma^{\mathcal{G}}$, leading to a nonlinear fixed-point equation and an iterative algorithm with favorable convergence properties. Before presenting the general algorithm, we illustrate its application in two simple examples.

*Example 3:* Consider the MPM assignment problem in Example 2, assuming $N = 2$ and distribution $p(x, y)$ is defined by positive-valued parameters $\alpha$, $\beta_1$ and $\beta_2$ as follows:

$$p(x) \propto \begin{cases} 1 & , & x_1 = x_2 \\ \alpha & , & x_1 \neq x_2 \end{cases} \quad \text{and} \quad p(y|x) = \prod_{n=1}^{N} \frac{1}{\sqrt{2\pi}} \exp\left[ -\frac{(y_n - \beta_n x_n)^2}{2} \right].$$

Note that $X_1$ and $X_2$ are marginally uniform and $\alpha$ captures their correlation (positive, zero, or negative when $\alpha$ is less than, equal to, or greater than unity, respectively), while $Y$ captures the presence of additive white Gaussian noise with signal-to-noise ratio at node $n$ equal to $\beta_n$. The (unconstrained) MPM strategy $\bar{\gamma}^*$ simplifies to a pair of threshold rules

$$L_1(y_1) \overset{u_1 = 1}{\underset{u_1 = 0}{\gtrless}} \bar{\eta}_1^* = \frac{1 + \alpha L_2(y_2)}{\alpha + L_2(y_2)} \quad \text{and} \quad L_2(y_2) \overset{u_2 = 1}{\underset{u_2 = 0}{\gtrless}} \bar{\eta}_2^* = \frac{1 + \alpha L_1(y_1)}{\alpha + L_1(y_1)},$$

where $L_n(y_n) = \exp\left[ \beta_n \left( y_n - \beta_n/2 \right) \right]$ denotes the *likelihood-ratio* local to node $n$. Let $\mathcal{E} = \{(1, 2)\}$ and define two network-constrained strategies: *myopic* strategy $\gamma^0$ employs thresholds $\eta_1^0 = \eta_2^0 = 1$, meaning each node $n$ acts to minimize $\Pr[U_n \neq X_n]$ as if in isolation, whereas *heuristic* strategy $\gamma^h$ employs thresholds $\eta_1^h = \eta_1^0$ and $\eta_2^h = \alpha^{2u_1 - 1}$, meaning node 2 adjusts its threshold as if $X_1 = u_1$ (i.e., as if the myopic decision by node 1 is always correct). Figure 1 compares these strategies and a pbp-optimal strategy $\gamma^k$—only $\gamma^k$ is both feasible and consistently "hedging" against all uncertainty i.e., $J(\gamma^0) \geq J(\gamma^k) \geq J(\bar{\gamma}^*)$.

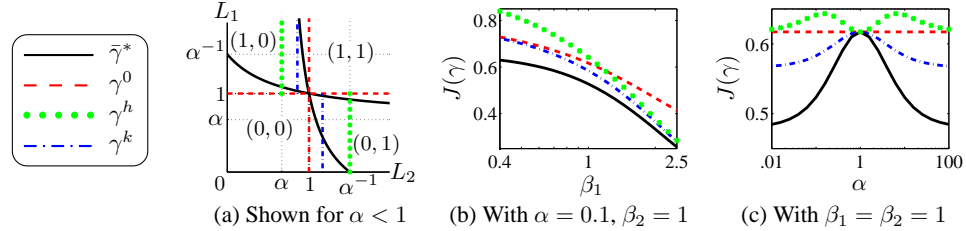

(a) Shown for $\alpha < 1$  (b) With $\alpha = 0.1$, $\beta_2 = 1$  (c) With $\beta_1 = \beta_2 = 1$

Figure 1. Comparison of the four alternative strategies in Example 3: (a) sketch of the decision regions in likelihood-ratio space, showing that network-constrained threshold rules cannot exactly reproduce $\bar{\gamma}^*$ (unless $\alpha = 1$); (b) bit-error-rate versus $\beta_1$ with $\alpha$ and $\beta_2$ fixed, showing $\gamma^h$ performs comparably to $\gamma^k$ when $Y_1$ is accurate relative to $Y_2$ but otherwise performs worse than even $\gamma^0$ (which requires no communication); (c) bit-error-rate versus $\alpha$ with $\beta_1$ and $\beta_2$ fixed, showing $\gamma^k$ uses the allotted bit of communication such that roughly 35% of the loss $J(\gamma^0) - J(\bar{\gamma}^*)$ is recovered.

*Example 4:* Extend Example 3 to $N > 2$ nodes, but assuming $X$ is equally-likely to be all zeros or all ones (i.e., the extreme case of positive correlation) and $Y$ has identically-accurate components with $\beta_n = 1$ for all $n$. The MPM strategy employs thresholds $\bar{\eta}_n^* = \prod_{i \in \mathcal{V} \setminus n} 1/L_i(y_i)$ for all $n$, leading to $U = \bar{\gamma}^*(Y)$ also being all zeros or all ones; thus, its *cost distribution*, or the probability mass function for $c(\bar{\gamma}^*(Y), X)$, has mass only on the values 0 and $N$. The myopic strategy employs thresholds $\eta_n^0 = 1$ for all $n$, leading to independent and identically-distributed (binary-valued) random variables $c_n(\gamma_n^0(Y_n), X_n)$; thus, its cost distribution, approaching a normal shape as $N$ gets large, has mass on all values $0, 1, \ldots, N$. Figure 2 considers a particular directed network $\mathcal{G}$ and, initializing to $\gamma^0$, shows the sequence of cost distributions resulting from the iterative offline algorithm—note the shape progression towards the cost distribution of the (infeasible) MPM strategy and the successive reduction in bit-error-rate $J(\gamma^k)$. Also noteworthy is the rapid convergence and the successive reduction in word-error-rate $\Pr[c(\gamma^k(Y), X) \neq 0]$.

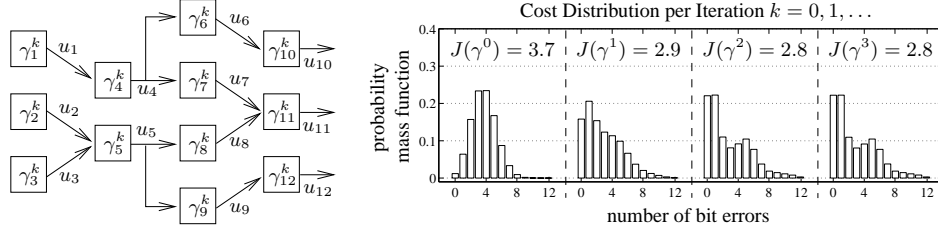

Figure 2. Illustration of the iterative offline computation given $p(x, y)$ as described in Example 4 and the directed network shown ($N = 12$). A Monte-Carlo analysis of $\bar{\gamma}^*$ yields an estimate for its bit-error-rate of $J(\bar{\gamma}^*) \approx 0.49$ (with standard deviation of 0.05)—thus, with a total of just $|\mathcal{E}| = 11$ bits of communication, the pbp-optimal strategy $\gamma^3$ recovers roughly 28% of the loss $J(\gamma^0) - J(\bar{\gamma}^*)$.

### 3.1 Necessary Optimality Conditions

We start by providing an explicit probabilistic interpretation of the general problem in (1).

**Lemma 1** *The minimum penalty $J(\gamma^*)$ defined in (1) is, firstly, achievable by a deterministic[1] strategy and, secondly, equivalently defined by*

$$J(\gamma^*) = \min_{p(u|y)} \sum_{x \in \{0,1\}^N} p(x) \sum_{u \in \{0,1\}^N} c(u, x) \int_{y \in \mathbb{R}^N} p(u|y)p(y|x)\, dy$$

$$\text{subject to} \quad p(u|y) = \prod_{n \in \mathcal{V}} p(u_n|y_n, u_{\pi(n)}).$$

Lemma 1 is primarily of conceptual value, establishing a correspondence between fixing a component rule $\gamma_n \in \Gamma_n$ and inducing a decision process $U_n$ from the information $(Y_n, U_{\pi(n)})$ local to node $n$. The following assumption permits analytical progress towards a finite parameterization for each function space $\Gamma_n$ and the basis of an offline algorithm.

**Assumption 1** *The observation process $Y$ satisfies $p(y|x) = \prod_{n \in \mathcal{V}} p(y_n|x)$.*

**Lemma 2** *Let Assumption 1 hold. Upon fixing a deterministic rule $\gamma_n \in \Gamma_n$ local to node $n$ (in correspondence with $p(u_n|y_n, u_{\pi(n)})$ by virtue of Lemma 1), we have the identity*

$$p(u_n|x, u_{\pi(n)}) = \int_{y_n \in \mathbb{R}} p(u_n|y_n, u_{\pi(n)})p(y_n|x)\, dy_n. \tag{4}$$

*Moreover, upon fixing a deterministic strategy $\gamma \in \Gamma^{\mathcal{G}}$, we have the identity*

$$p(u|x) = \prod_{n \in \mathcal{V}} p(u_n|x, u_{\pi(n)}). \tag{5}$$

Lemma 2 implies fixing component rule $\gamma_n \in \Gamma_n$ is in correspondence with inducing the conditional distribution $p(u_n|x, u_{\pi(n)})$, now a probabilistic description that persists local to node $n$ no matter the rule $\gamma_i$ at any other node $i \in \mathcal{V}\backslash n$. Lemma 2 also introduces further structure in the constrained optimization expressed by Lemma 1: recognizing the integral over $\mathbb{R}^N$ to equal $p(u|x)$, (4) and (5) together imply it can be expressed as a product of

component integrals, each over $\mathbb{R}$. We now argue that, despite these simplifications, the component rules of $\gamma^*$ continue to be globally coupled.

Starting with any deterministic strategy $\gamma \in \Gamma^{\mathcal{G}}$, consider optimizing the $n$th component rule $\gamma_n$ over $\Gamma_n$ assuming all other components stay fixed. With $\gamma_n$ a degree-of-freedom, decision process $U_n$ is no longer well-defined so each $u_n \in \{0,1\}$ merely represents a candidate decision local to node $n$. Online, each local decision will be made only upon receiving both the local observation $Y_n = y_n$ and all parents' local decisions $U_{\pi(n)} = u_{\pi(n)}$. It follows that node $n$, upon deciding a particular $u_n$, may assert that random vector $U$ is restricted to values in the subset $\mathcal{U}[u_{\pi(n)}, u_n] = \{u' \in \{0,1\}^N \mid u'_{\pi(n)} = u_{\pi(n)}, u'_n = u_n\}$. Then, viewing $(Y_n, U_{\pi(n)})$ as a composite local observation and proceeding in the manner by which (3) is derived, the pbp-optimal relaxation of (1) reduces to the following form.

**Proposition 1** *Let Assumption 1 hold. In an optimal network-constrained strategy $\gamma^* \in \Gamma^{\mathcal{G}}$, for each $n$ and assuming all components $i \in \mathcal{V}\backslash n$ are fixed at rules $\gamma_i^*$ (each in correspondence with $p^*(u_i|x, u_{\pi(i)})$ by virtue of Lemma 2), the rule $\gamma_n^*$ satisfies*

$$\gamma_n^*(Y_n, U_{\pi(n)}) = \arg \min_{u_n \in \{0,1\}} \sum_{x \in \{0,1\}^N} b_n^*(u_n, x; U_{\pi(n)}) p(Y_n|x) \quad \text{with probability one}$$

(6)

*where, for each $u_{\pi(n)} \in \{0,1\}^{|\pi(n)|}$,*

$$b_n^*(u_n, x; u_{\pi(n)}) = p(x) \sum_{u \in \mathcal{U}[u_{\pi(n)}, u_n]} c(u, x) \prod_{i \in \mathcal{V}\backslash n} p^*(u_i|x, u_{\pi(i)}). \tag{7}$$

Of note are (i) the likelihood function $p(Y_n|x)$ is a finite-dimensional sufficient statistic of $Y_n$, (ii) real-valued coefficients $b_n$ provide a finite parameterization of the function space $\Gamma_n$ and (iii) the pbp-optimal coefficient values $b_n^*$, while still computable offline, also depend on the distributions $p^*(u_i|x, u_{\pi(i)})$ in correspondence with all fixed rules $\gamma_i^*$.

### 3.2 Offline Message-Passing Algorithm

Let $f_n$ map from coefficients $\{b_i; i \in \mathcal{V}\backslash n\}$ to coefficients $b_n$ by the following operations:

1. for each $i \in \mathcal{V}\backslash n$, compute $p(u_i|x, u_{\pi(i)})$ via (4) and (6) given $b_i$ and $p(y_i|x)$;
2. compute $b_n$ via (7) given $p(x)$, $c(u, x)$ and $\{p(u_i|x, u_{\pi(i)}); i \in \mathcal{V}\backslash n\}$.

Then, the simultaneous satisfaction of Proposition 1 at all $N$ nodes can be viewed as a system of $2^{N+1} \sum_{n \in \mathcal{V}} 2^{|\pi(n)|}$ nonlinear equations in as many unknowns,

$$b_n = f_n(b_1, \ldots, b_{n-1}, b_{n+1}, \ldots, b_N), \quad n = 1, \ldots, N, \tag{8}$$

or, more concisely, $b = f(b)$. The connection between each $f_n$ and Proposition 1 affords an equivalence between solving the fixed-point equation $f$ via a Gauss-Seidel iteration and minimizing $J(\gamma)$ via a coordinate-descent iteration [9], implying an algorithm guaranteed to terminate and achieve penalty no greater than that of an arbitrary initial strategy $\gamma^0 \leftrightarrow b^0$.

**Proposition 2** *Initialize to any coefficients $b^0 = (b_1^0, \ldots, b_N^0)$ and generate the sequence $\{b^k\}$ using a component-wise iterative application of $f$ in (8) i.e., for $k = 1, 2, \ldots$,*

$$b_n^k := f_n(b_1^{k-1}, \ldots, b_{n-1}^{k-1}, b_{n+1}^k, \ldots, b_N^k), \quad n = N, N-1, \ldots, 1. \tag{9}$$

*If Assumption 1 holds, the associated sequence $\{J(\gamma^k)\}$ is non-increasing and converges:*

$$J(\gamma^0) \geq J(\gamma^1) \geq \cdots \geq J(\gamma^k) \to J^* \geq J(\gamma^*) \geq J(\bar{\gamma}^*).$$

Direct implementation of (9) is clearly imprudent from a computational perspective, because the transformation from fixed coefficients $b_n^k$ to the corresponding distribution $p^k(u_n|x, u_{\pi(n)})$ need not be repeated within every component evaluation of $f$. In fact, assuming every node $n$ stores in memory its own likelihood function $p(y_n|x)$, this transformation can be accomplished locally (cf. (4) and (6)) and, also assuming the resulting distribution is broadcast to all other nodes before they proceed with their subsequent component evaluation of $f$, the termination guarantee of Proposition 2 is retained. Requiring every node to perform a network-wide broadcast within every iteration $k$ makes (9) a decidedly global algorithm, not to mention that each node $n$ must also store in memory $p(x, y_n)$ and $c(u, x)$ to carry forth the supporting local computations.

**Assumption 2** *The cost function satisfies $c(u, x) = \sum_{n \in \mathcal{V}} c_n(u_n, x)$ for some collection of functions $\{c_n : \{0, 1\}^{N+1} \to \mathbb{R}\}$ and the directed graph $\mathcal{G}$ is tree-structured.*

**Proposition 3** *Under Assumption 2, the following two-pass procedure is identical to (9):*

- *Forward-pass at node $n$: upon receiving messages from all parents $i \in \pi(n)$, store them for use in the next reverse-pass and send to each child $j \in \chi(n)$ the following messages:*

$$P_{n \to j}^k(u_n|x) := \sum_{u_{\pi(n)} \in \{0,1\}^{|\pi(n)|}} p^{k-1}(u_n|x, u_{\pi(n)}) \prod_{i \in \pi(n)} P_{i \to n}^k(u_i|x). \quad (10)$$

- *Reverse-pass at node $n$: upon receiving messages from all children $j \in \chi(n)$, update*

$$b_n^k(u_n, x; u_{\pi(n)}) := p(x) \prod_{i \in \pi(n)} P_{i \to n}^k(u_i|x) \left( c_n(u_n, x) + \sum_{j \in \chi(n)} C_{j \to n}^k(u_n, x) \right) \quad (11)$$

*and the corresponding distribution $p^k(u_n|x, u_{\pi(n)})$ via (4) and (6), store the distribution for use in the next forward pass and send to each parent $i \in \pi(n)$ the following messages:*

$$C_{n \to i}^k(u_i, x) := \sum_{u_n \in \{0,1\}} p(u_n|x, u_i) \left( c_n(u_n, x) + \sum_{j \in \chi(n)} C_{j \to n}^k(u_n, x) \right), \quad (12)$$

$$p(u_n|x, u_i) = \sum_{u_{\pi(n)} \in \{u' \in \{0,1\}^{|\pi(n)|} | u_i' = u_i\}} p^k(u_n|x, u_{\pi(n)}) \prod_{\ell \in \pi(n) \setminus i} P_{\ell \to n}^k(u_\ell|x).$$

An intuitive interpretation of Proposition 3, from the perspective of node $n$, is as follows. From (10) in the forward pass, the messages received from each parent define what, during subsequent online operation, that parent's local decision means (in a likelihood sense) about its ancestors' outputs and the hidden process. From (12) in the reverse pass, the messages received from each child define what the local decision will mean (in an expected cost sense) to that child and its descendants. From (11), both types of incoming messages impact the local rule update and, in turn, the outgoing messages to both types of neighbors. While Proposition 3 alleviates the need for the iterative global broadcast of distributions $p^k(u_n|x, u_{\pi(n)})$, the explicit dependence of (10)-(12) on the full vector $x$ implies the memory and computation requirements local to each node can still be exponential in $N$.

**Assumption 3** *The hidden process $X$ is Markov on $\mathcal{G}$, or $p(x) = \prod_{n \in \mathcal{V}} p(x_n|x_{\pi(n)})$, and all component likelihoods/costs satisfy $p(y_n|x) = p(y_n|x_n)$ and $c_n(u_n, x) = c_n(u_n, x_n)$.*

**Proposition 4** *Under Assumption 3, the iterates in Proposition 3 specialize to the form of*

$$b_n^k(u_n, x_n; u_{\pi(n)}), \quad P_{n \to j}^k(u_n|x_n) \quad and \quad C_{n \to i}^k(u_i, x_i), \qquad k = 0, 1, \dots$$

*and each node $n$ need only store in memory $p(x_{\pi(n)}, x_n, y_n)$ and $c_n(u_n, x_n)$ to carry forth the supporting local computations. (The actual equations can be found in [8].)*

Proposition 4 implies the convergence properties of Proposition 2 are upheld with maximal efficiency (linear in $N$) when $\mathcal{G}$ is tree-structured and the global distribution and costs satisfy $p(x,y) = \prod_{n \in \mathcal{V}} p(x_n | x_{\pi(n)}) p(y_n | x_n)$ and $c(u, x) = \sum_{n \in \mathcal{V}} c_n(u_n, x_n)$, respectively. Note that these conditions hold for the MPM assignment problems in Examples 3 & 4.

## 4  Discussion

Our decision-theoretic variational approach reflects several departures from existing methods for communication-constrained inference. Firstly, instead of imposing the constraints on an algorithm derived from an ideal model, we explicitly model the constraints and derive a different algorithm. Secondly, our penalty function drives the approximation by the desired application of inference (e.g., posterior assignment) as opposed to a generic error measure on the result of inference (e.g., divergence in true and approximate marginals). Thirdly, the necessary offline computation gives rise to a downside, namely less flexibility against time-varying statistical environments, decision objectives or network conditions.

Our development also evokes principles in common with other research areas. Similar to the sum-product version of Belief Propagation (BP), our message-passing algorithm originates assuming a tree structure, an additive cost and a synchronous message schedule. It is thus enticing to claim that the maturation of BP (e.g., max-product, asynchronous schedule, cyclic graphs) also applies, but unique aspects to our development (e.g., directed graph, weak convergence, asymmetric messages) merit caution. That we solve for correlated equilibria and depend on probabilistic structure commensurate with cost structure for efficiency is in common with graphical games [10], which distinctly are formulated on undirected graphs and absent of hidden variables. Finally, our offline computation resembles learning a conditional random field [11], in the sense that factors of $p(u|x)$ are iteratively modified to reduce penalty $J(\gamma)$; online computation via strategy $u = \gamma(y)$, repeated per realization $Y = y$, is then viewed as sampling from this distribution. Along the learning thread, a special case of our formulation appears in [12], but assuming $p(x, y)$ is unknown.

### Acknowledgments

This work supported by the Air Force Office of Scientific Research under contract FA9550-04-1 and by the Army Research Office under contract DAAD19-00-1-0466. We are grateful to Professor John Tsitsiklis for taking time to discuss the correctness of Proposition 1.

## Footnotes

[1]A randomized (or mixed) strategy, modeled as a probabilistic selection from a finite collection of deterministic strategies, takes more inputs than just the observation process $Y$. That deterministic strategies suffice, however, justifies "post-hoc" our initial abuse of notation for elements in the set $\Gamma$.

### References

[1]  J. Pearl. *Probabilistic Reasoning in Intelligent Systems*. Morgan Kaufmann, 1988.

[2]  L. Chen, et al. Data association based on optimization in graphical models with application to sensor networks. *Mathematical and Computer Modeling*, 2005. To appear.

[3]  A. T. Ihler, et al. Message errors in belief propagation. *Advances in NIPS 17*, MIT Press, 2005.

[4]  M. I. Jordan, et al. An introduction to variational methods for graphical models. *Learning in Graphical Models*, pp. 105–161, MIT Press, 1999.

[5]  J. N. Tsitsiklis. Decentralized detection. *Adv. in Stat. Sig. Proc.*, pp. 297–344, JAI Press, 1993.

[6]  P. K. Varshney. *Distributed Detection and Data Fusion*. Springer-Verlag, 1997.

[7]  J. Marschak and R. Radner. *The Economic Theory of Teams*. Yale University Press, 1972.

[8]  O. P. Kreidl and A. S. Willsky. Posterior assignment in directed graphical models with minimal online communication. Available: `http://web.mit.edu/opk/www/res.html`

[9]  D. P. Bertsekas. *Nonlinear Programming*. Athena Scientific, 1995.

[10]  S. Kakade, et al. Correlated equilibria in graphical games. *ACM-CEC*, pp. 42–47, 2003.

[11]  J. Lafferty, et al. Conditional random fields: Probabilistic models for segmenting and labeling sequence data. *ICML*, 2001.

[12]  X. Nguyen, et al. Decentralized detection and classification using kernel methods. *ICML*,2004.
